# Modeling the Modulatory Effect of Attention on Human Spatial Vision

**Laurent Itti**
Computer Science Department, Hedco Neuroscience Building HNB-30A,
University of Southern California, Los Angeles, CA 90089-2520, U.S.A.

**Jochen Braun**
nstitute of Neuroscience and School of Computing,
University of Plymouth, Plymouth Devon PL4 8AA, U.K.

**Christof Koch**
Computation and Neural Systems Program, MC 139-74,
California Institute of Technology, Pasadena, CA 91125, U.S.A.

## Abstract

We present new simulation results, in which a computational model of interacting visual neurons simultaneously predicts the modulation of spatial vision thresholds by focal visual attention, for five dual-task human psychophysics experiments. This new study complements our previous findings that attention activates a winner-take-all competition among early visual neurons within one cortical hypercolumn. This "intensified competition" hypothesis assumed that attention equally affects all neurons, and yielded two single-unit predictions: an increase in gain and a sharpening of tuning with attention. While both effects have been separately observed in electrophysiology, no single-unit study has yet shown them simultaneously. Hence, we here explore whether our model could still predict our data if attention might only modulate neuronal gain, but do so non-uniformly across neurons and tasks. Specifically, we investigate whether modulating the gain of only the neurons that are loudest, best-tuned, or most informative about the stimulus, or of all neurons equally but in a task-dependent manner, may account for the data. We find that none of these hypotheses yields predictions as plausible as the intensified competition hypothesis, hence providing additional support for our original findings.

## 1  INTRODUCTION

Psychophysical studies as well as introspection indicate that we are not blind outside the focus of attention, and that we can perform simple judgments on objects not being attended to [1], though those judgments are less accurate than in the

presence of attention [2, 3]. While attention thus appears not to be mandatory for early vision, there is mounting experimental evidence from single-neuron electrophysiology [4, 5, 6, 7, 8, 9, 10], human psychophysics [11, 12, 13, 14, 3, 2, 15, 16] and human functional imaging experiments [17, 18, 19, 20, 21, 22, 23] that focal visual attention modulates, top-down, activity in early sensory processing areas. In the visual domain, this modulation can be either spatially-defined (i.e., neuronal activity only at the retinotopic location attended to is modulated) or feature-based (i.e., neurons with stimulus preference matching the stimulus attended to are enhanced throughout the visual field), or a combination of both [7, 10, 24].

Computationally, the modulatory effect of attention has been described as enhanced gain [8, 10], biased [4] or intensified [14, 2] competition, enhanced spatial resolution [3], sharpened neuronal tuning [5, 25] or as modulated background activity [19], effective stimulus strength [26] or noise [15]. One theoretical difficulty in trying to understand the modulatory effect of attention in computational terms is that, although attention profoundly alters visual perception, it is not equally important to all aspects of vision. While electrophysiology demonstrates "increased firing rates" with attention for a given task, psychophysics show "improved discrimination thresholds" on some other tasks, and functional magnetic resonance imaging (fMRI) reports "increased activation" for yet other tasks, the computational mechanism at the origin of these observations remains largely unknown and controversial.

While most existing theories are associated to a specific body of data, and a specific experimental task used to engage attention in a given experiment, we have recently proposed a unified computational account [2] that spans five such tasks (32 thresholds under two attentional conditions, i.e., 64 datapoints in total). This theory predicts that attention activates a winner-take-all competition among neurons tuned to different orientations within a single hypercolumn in primary visual cortex (area V1). It is rooted in new information-theoretic advances [27], which allowed us to quantitatively relate single-unit activity in a computational model to human psychophysical thresholds. A consequence of our "intensified competition hypothesis" is that attention both increases the gain of early visual neurons (by a factor 3.3), and sharpens their tuning for the orientation (by 40%) and spatial frequency (by 30%). While gain modulation has been observed in some of the single-unit studies mentioned above [8, 10] (although much smaller effects are typically reported, on the order of 10-15%, probably because these studies do not use dual-task paradigms and thus poorly engage the attention of the animal towards or away from the stimulus of interest), and tuning modulation has been observed in other single-unit studies [5, 25], both gain and tuning modulation have not been simultaneously observed in a single electrophysiological set of experiments [10].

In the present study, we thus investigate alternatives to our intensified competition hypothesis which only involve gain modulation. Our previous results [2] have shown that both increased gain and sharper tuning were necessary to simultaneously account for our five pattern discrimination tasks, if those modulatory effects were to equally affect all visual neurons at the location of the stimulus and to be equal for all tasks. Thus, we here extend our computational search space under two new hypotheses: First, we investigate whether attention might only modulate the gain of selected sub-populations of neurons (responding the loudest, best tuned, or most informative about the stimulus) in a task-independent manner. Second, we investigate whether attention might equally modulate the gain of all visual neurons responding to the stimulus, but in a task-dependent manner. Thus, the goal of the present study is to determine, using new computational simulations, whether the modulatory effect of attention on early visual processing might be explained by gain-only modulations, if such modulations are allowed to be sufficiently complex

(affecting only select visual neurons, or task-dependent). Although attention certainly affects most stages of visual processing, we here continue to focus on early vision, as it is widely justified by electrophysiological and fMRI evidence that some modulation does happen very early in the processing hierarchy [5, 8, 9, 23].

## 2 PSYCHOPHYSICAL DATA

Our recent study [2] measured psychophysical thresholds for three pattern discrimination tasks (contrast, orientation and spatial frequency discriminations), and two spatial masking tasks (32 thresholds). We used a dual-task paradigm to measure thresholds either when attention was fully available to the task of interest (presented in the near periphery), or when it was poorly available because engaged elsewhere by a concurrent attention-demanding task (a letter discrimination task at the center of the display). The results are summarized in **Fig. 1** and [2].

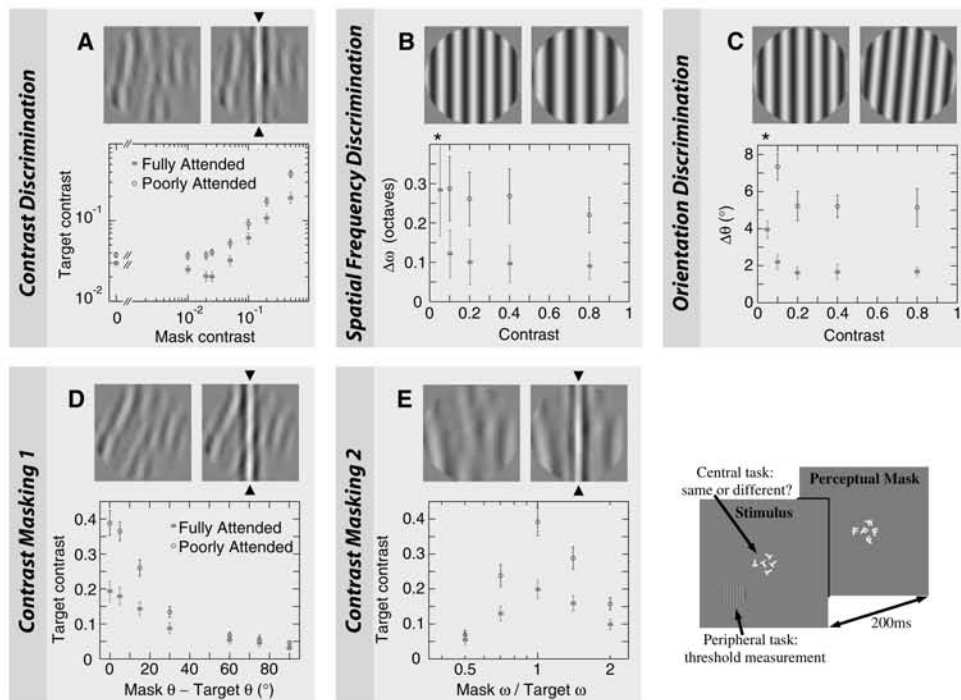

Figure 1: Psychophysical data from Lee *et al.* Central targets appeared at $0 - 0.8°$ eccentricity and measured $0.4°$ across. Peripheral targets appeared for 250 ms at $4°$ eccentricity, in a circular aperture of $1.5°$. They were either sinusoidal gratings (**B, C**) or vertical stripes whose luminance profile was given by the 6th derivative of a Gaussian (**A, D, E**). Mask patterns were generated by superimposing 100 Gabor filters, positioned randomly within the circular aperture (**A, D, E**). Thresholds were established with an adaptive staircase method (80 trials per block). A complex pattern of effects is observed, with a strong modulation of orientation and spatial frequency discriminations (**B, C**), smaller modulation of contrast discriminations (**A**), and modulation of contrast masking that depends on stimulus configurations (**D, E**). These complex observations can be simultaneously accounted for by our computational model of one hypercolumn in primary visual cortex.

# 3 COMPUTATIONAL MODEL

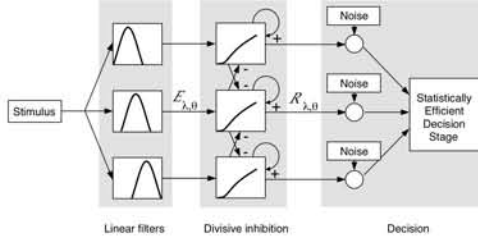

Linear filters    Divisive inhibition      Decision

The model developed to quantitatively account for this data comprises three successive stages [14, 27]. In the first stage, a bank of Gabor-like linear filters (12 orientations and 5 spatial scales) analyzes a given visual location, similarly to a cortical hypercolumn. In the second stage, filters nonlinearly interact through both a self-excitation component, and a divisive inhibition component that is derived from a pool of similarly-tuned units. With $E_{\lambda,\theta}$ being the linear response from a unit tuned to spatial period $\lambda$ and orientation $\theta$, the response $R_{\lambda,\theta}$ after interactions is given by (see [27] for additional details):

$$R_{\lambda,\theta} = \frac{(A.E_{\lambda,\theta})^{\gamma}}{(S)^{\delta} + \sum_{(\lambda',\theta') \in \Lambda \times \Theta} W_{\lambda,\theta}(\lambda',\theta')\,(A.E_{\lambda',\theta'})^{\delta}} + B, \qquad (1)$$

where: 
$$W_{\lambda,\theta}(\lambda',\theta') = \exp\left(-\frac{(\log(\lambda') - \log(\lambda))^2}{2\Lambda_{\lambda}^2} - \frac{(\theta' - \theta)^2}{2\Lambda_{\theta}^2}\right) \qquad (2)$$

is a 2D Gaussian weighting function centered around $(\lambda,\theta)$ whose widths are determined by the scalars $\Lambda_{\theta}$ and $\Lambda_{\lambda}$. The neurons are assumed to be noisy, with noise variance $V_{\lambda,\theta}^2$ given by a generalized Poisson model: $V_{\lambda,\theta}^2 = \beta(R_{\lambda,\theta} + \epsilon)$.

The third stage relates activity in the population of interacting noisy units to behavioral discrimination performance. To allow us to quantitatively predict thresholds from neural activity for any task, our decision stage assumes that observers perform close to an *unbiased efficient statistic*, that is, the best possible estimator (in the statistical estimation sense) of the characteristics of the stimulus given the noisy neuronal responses. This methodology (described further in [27]) allows us to quantitatively compute thresholds in any behavioral situation, and eliminates the need for task-dependent assumptions about the decision strategy used by the observers.

# 4 RESULTS and DISCUSSION

The 10 free model parameters (**Fig. 2**) were automatically adjusted to best fit the psychophysical data from all experiments, using a multidimensional downhill simplex with simulated annealing overhead (see [27]), running on our 16-CPU Linux Beowulf system (16 × 733 MHz, 4 GB RAM, 0.5 TB disk; see http://iLab.usc.edu/beo/). Parameters were simultaneously adjusted for both attentional conditions; that is, the total fit error was the sum of the error obtained with the baseline set of parameters on the poorly attended data, and of the error obtained with the same parameters plus some attentional perturbation on the fully attended data. Thus, no bias was given to any of the two attentional conditions.

For the "separate fits" (**Fig. 2**), all parameters were allowed to differ with attention [2], while only the interaction parameters ($\gamma, \delta$) could differ in the "intensified competition" case. The "loudest filter" was the one responding loudest to the entire visual pattern presented (stimulus + mask), the "best-tuned filter" was that responding best to the stimulus component alone, and the "most informative filter" was that for which the Fisher information about the stimulus was highest (see

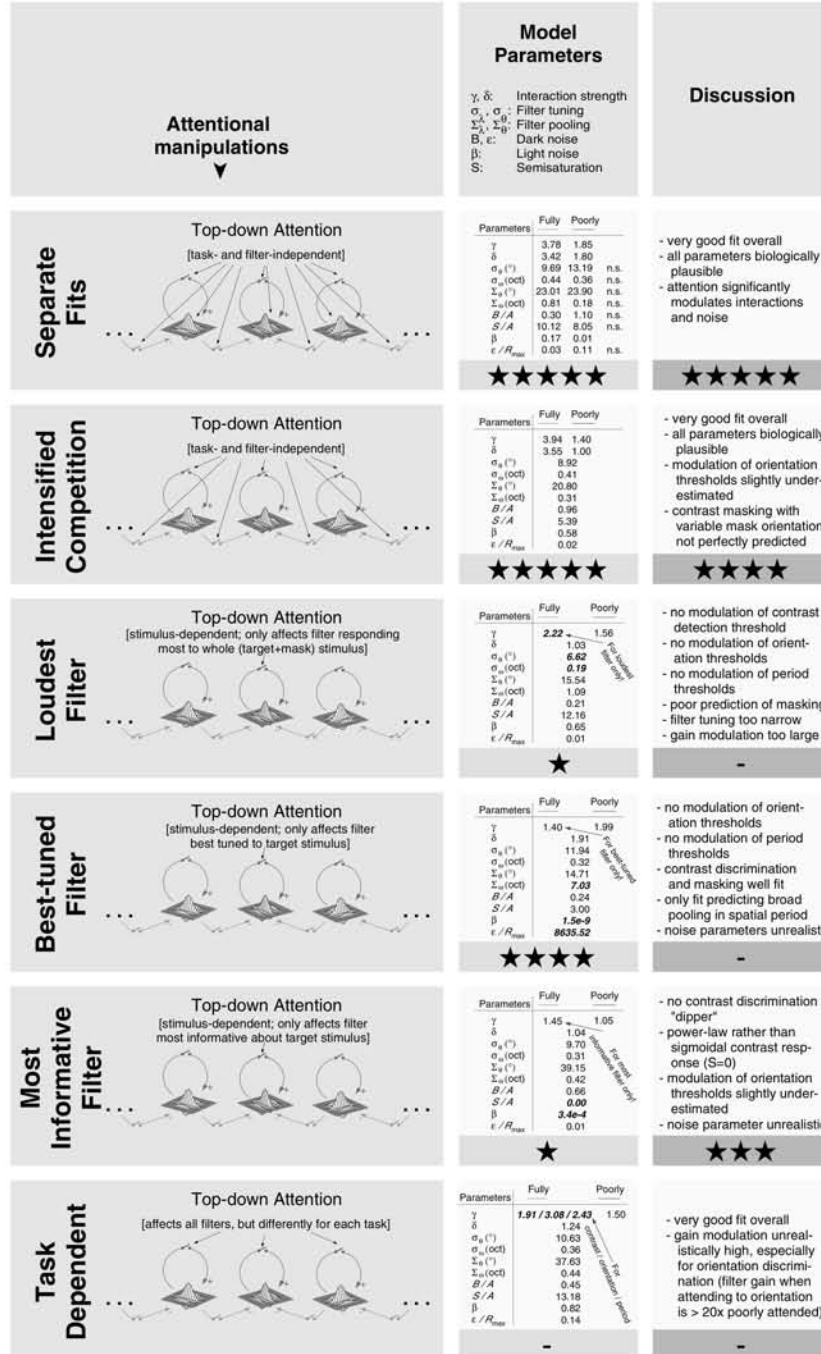

Figure 2: Attentional modulation hypotheses and corresponding model parameters. See next page for the corresponding model predictions on our five tasks, for the hypotheses shown. The middle column shows which parameters were allowed to differ with attention, and the best-fit values for both attentional conditions.

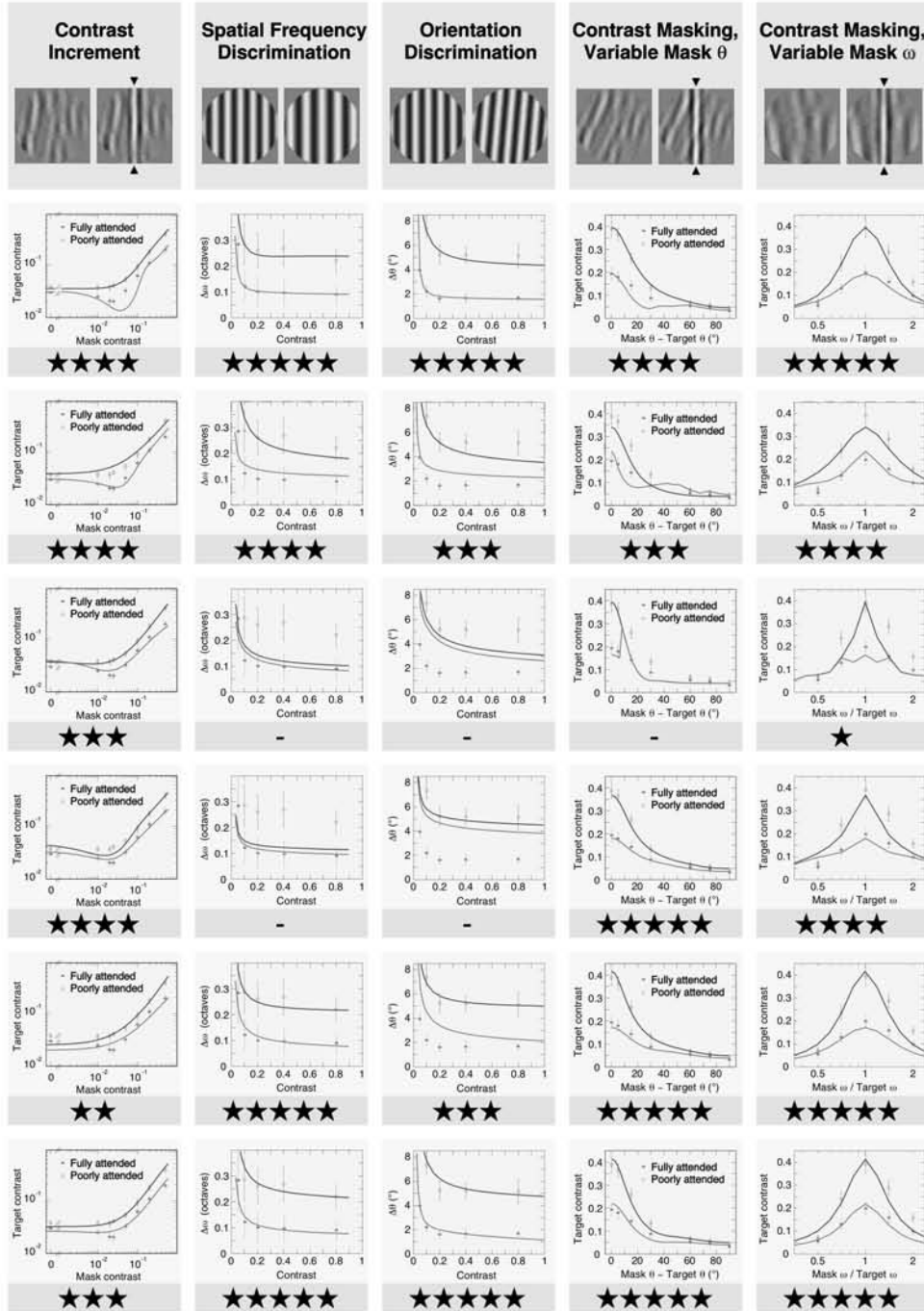

Figure 3: Model predictions for the different attentional modulation hypotheses studied. The different rows correspond to the different attentional manipulations studied, as labeled in the previous figure. Ratings (stars below the plots) were derived from the residual error of the fits.

[14, 27]). Finally, in the "task-dependent" case, the gain of all filters was affected equally (parameter $\gamma$), but with three different values for the contrast (discrimination and masking), orientation and spatial frequency tasks. Overall, very good fits were obtained in the "separate fits" and "intensified competition" conditions (as previously reported), as well as in the "most informative filter" and "task-dependent" conditions **(Fig. 3)**, while the two remaining hypotheses yielded very poor predictions of orientation and spatial frequency discriminations. In the "most informative filter" case, the dipper in the contrast increment thresholds was missing because the nonlinear response function of the neurons converged to a power law rather than the usually observed sigmoid [27]; thus, this hypothesis lost some of its appeal because of its lower biological plausibility. More importantly, a careful analysis of the very promising results for the "task-dependent" case also revealed their low biological plausibility, with a gain modulation in excess of 20-fold being necessary to explain the orientation discrimination data **(Fig. 2)**.

In summary, we found that none of the simpler (gain only) attentional manipulations studied here could explain as well the psychophysical data as our previous manipulation, "intensified competition," which implied that attention both increases the gain and sharpens the tuning of early visual neurons. Two of the four new manipulations studied yielded good quantitative model predictions: affecting the gain of the filter most informative about the target stimulus, and affecting the gain of all filters in a task-dependent manner. In both cases, however, some of the internal model parameters associated with the fits were biologically unrealistic, thus reducing the plausibility of these two hypotheses. In all manipulations studied, the greatest difficulty was in trying to account for the orientation and spatial frequency discrimination data without unrealistically high gain changes (greater than 20-fold). Our results hence provide additional evidence for the hypothesis that sharpening of tuning may be necessary to account for these thresholds, as was originally suggested by our separate fits and our intensified competition hypothesis and has been recently supported by new investigations [16].

## Acknowledgements

This research was supported by the National Eye Institute, the National Science Foundation, the NSF-supported ERC center at Caltech, the National Institutes for Mental Health, and startup funds from the Charles Lee Powell Foundation and the USC School of Engineering.

## References

[1] Braun J & Sagi D. *Percept Psychophys*, 1990;**48**(1):45–58.

[2] Lee DK, Itti L, Koch C *et al*. *Nat Neurosci*, 1999;**2**(4):375–81.

[3] Yeshurun Y & Carrasco M. *Nature*, 1998;**396**(6706):72–75.

[4] Moran J & Desimone R. *Science*, 1985;**229**(4715):782–4.

[5] Spitzer H, Desimone R & Moran J. *Science*, 1988;**240**(4850):338–40.

[6] Chelazzi L, Miller EK, Duncan J *et al*. *Nature*, 1993;**363**(6427):345–7.

[7] Motter BC. *J Neurosci*, 1994;**14**(4):2178–89.

[8] Treue S & Maunsell JH. *Nature*, 1996;**382**(6591):539–41.

[9] Luck SJ, Chelazzi L, Hillyard SA *et al*. *J Neurophysiol*, 1997;**77**(1):24–42.

[10] Treue S & Trujillo JCM. *Nature*, 1999;**399**(6736):575–579.

[11] Nakayama K & Mackeben M. *Vision Res*, 1989;**29**(11):1631–47.

[12] Bonnel AM, Stein JF & Bertucci P. *Q J Exp Psychol A*, 1992;**44**(4):601–26.

[13] Lee DK, Koch C & Braun J. *Vision Res*, 1997;**37**(17):2409–18.

[14] Itti L, Braun J, Lee DK *et al*. In *NIPS*11*. MIT Press, 1999; pp. 789–795.

[15] Dosher BA & Lu ZL. *Vision Res*, 2000;**40**(10-12):1269–1292.

[16] Carrasco M, Penpeci-Talgar C & Eckstein M. *Vision Res*, 2000;**40**(10-12):1203–1215.

[17] Corbetta M, Miezin FM, Dobmeyer S *et al*. *Science*, 1990;**248**(4962):1556–9.

[18] Rees G, Frackowiak R & Frith C. *Science*, 1997;**275**(5301):835–8.

[19] Chawla D, Rees G & Friston KJ. *Nat Neurosci*, 1999;**2**(7):671–676.

[20] Brefczynski JA & DeYoe EA. *Nat Neurosci*, 1999;**2**(4):370–374.

[21] Corbetta M, Kincade JM, Ollinger JM *et al*. *Nat Neurosci*, 2000;**3**(3):292–297.

[22] Kanwisher N & Wojciulik E. *Nat Rev Neurosci*, 2000;**1**:91–100.

[23] Ress D, Backus BT & Heeger DJ. *Nat Neurosci*, 2000;**3**(9):940–945.

[24] Barcelo F, Suwazono S & Knight RT. *Nat Neurosci*, 2000;**3**(4):399–403.

[25] Desimone R & Duncan J. *Annu Rev Neurosci*, 1995;**18**:193–222.

[26] Reynolds JH, Pasternak T & Desimone R. *Neuron*, 2000;**26**(3):703–714.

[27] Itti L, Koch C & Braun J. *J Opt Soc Am A*, 2000;**17**(11):1899–1917.
